# OPTIMIZATION BY MEAN FIELD ANNEALING

Griff Bilbro
ECE Dept.
NCSU
Raleigh, NC 27695

Reinhold Mann
Eng. Physics and Math. Div.
Oak Ridge Natl. Lab.
Oak Ridge, TN 37831

Thomas K. Miller
ECE Dept.
NCSU
Raleigh, NC 27695

Wesley. E. Snyder
ECE Dept.
NCSU
Raleigh, NC 27695

David E. Van den Bout
ECE Dept.
NCSU
Raleigh, NC 27695

Mark White
ECE Dept.
NCSU
Raleigh, NC 27695

## ABSTRACT

Nearly optimal solutions to many combinatorial problems can be found using stochastic simulated annealing. This paper extends the concept of simulated annealing from its original formulation as a Markov process to a new formulation based on mean field theory. *Mean field annealing* essentially replaces the discrete degrees of freedom in simulated annealing with their average values as computed by the mean field approximation. The net result is that equilibrium at a given temperature is achieved 1-2 orders of magnitude faster than with simulated annealing. A general framework for the mean field annealing algorithm is derived, and its relationship to Hopfield networks is shown. The behavior of MFA is examined both analytically and experimentally for a generic combinatorial optimization problem: graph bipartitioning. This analysis indicates the presence of critical temperatures which could be important in improving the performance of neural networks.

## STOCHASTIC VERSUS MEAN FIELD

In combinatorial optimization problems, an objective function or *Hamiltonian*, $H(s)$, is presented which depends on a vector of interacting *spins*, $s = \{s_1, \ldots, s_N\}$, in some complex nonlinear way. Stochastic simulated annealing (SSA) (S. Kirkpatrick, C. Gelatt, and M. Vecchi (1983)) finds a global minimum of $H$ by combining gradient descent with a random process. This combination allows, under certain conditions, choices of s which actually *increase* $H$, thus providing SSA with a mechanism for escaping from local minima. The frequency and severity of these uphill moves is reduced by slowly decreasing a parameter $T$ (often referred to as the *temperature*) such that the system settles into a global optimum.

Two conceptual operations are involved in simulated annealing: a *thermostatic operation* which schedules decreases in the temperature, and a *relaxation operation*

which iteratively finds the equilibrium solution at the new temperature using the final state of the system at the previous temperature as a starting point. In SSA, relaxation occurs by randomly altering components of s with a probability determined by both $T$ and the change in $H$ caused by each such operation. This corresponds to probabilistic transitions in a Markov chain. In *mean field annealing* (MFA), some aspects of the optimization problem are replaced with their means or averages from the underlying Markov chain (e.g. s is replaced with its average, $\langle s \rangle$). As the temperature is decreased, the MFA algorithm updates these averages based on their values at the previous temperature. Because computation using the means attains equilibrium faster than using the corresponding Markov chain, MFA relaxes to a solution at each temperature much faster than does SSA, which leads to an overall decrease in computational effort.

In this paper, we present the MFA formulation in the context of the familiar Ising Hamiltonian and discuss its relationship to Hopfield neural networks. Then the application of MFA to the problem of graph bipartitioning is discussed, where we have analytically and experimentally investigated the affect of temperature on the behavior of MFA and observed speedups of 50:1 over SSA.

## MFA AND HOPFIELD NETWORKS

Optimization theory, like physics, often concerns itself with systems possessing a large number of interacting degrees of freedom. Physicists often simplify their problems by using the *mean field approximation*: a simple analytic approximation of the behavior of systems of particles or spins in thermal equilibrium. In a corresponding manner, arbitrary functions can be optimized by using an analytic version of stochastic simulated annealing based on a technique analogous to the mean field approximation. The derivation of MFA presented here uses the *naive mean field* (D. J. Thouless, P.W. Anderson, and R.G. Palmer (1977)) and starts with a simple Ising Hamiltonian of N spins coupled by a product interaction:

$$H(s) = \sum_i h_i s_i + \sum_i \sum_{j \neq i} V_{ij} s_i s_j \quad \text{where} \quad \begin{cases} V_{ij} = V_{ji} & symmetry \\ s_i \in \{0,1\} & integer\ spins. \end{cases}$$

Factoring $H(s)$ shows the interaction between a spin $s_i$ and the rest of the system:

$$H(s) = s_i \cdot \left( h_i + 2 \sum_{j \neq i} V_{ij} s_j \right) + \sum_{k \neq i} h_k s_k + \sum_{k \neq i} \sum_{j \neq k, i} V_{kj} s_k s_j. \quad (1)$$

The *mean* or *effective* field affecting $s_i$ is the average of its coefficient in (1):

$$\Phi_i = \langle h_i + 2 \sum_{j \neq i} V_{ij} s_j \rangle = h_i + 2 \sum_{j \neq i} V_{ij} \langle s_j \rangle = H|_{\langle s_i \rangle = 1} - H|_{\langle s_i \rangle = 0}. \quad (2)$$

The last part of (2) shows that, for the Ising case, the mean field can be simply calculated from the difference in the Hamiltonian caused by changing $\langle s_i \rangle$ from zero

1. Initialize spin averages and add noise: $s_i = 1/2 + \delta \ \forall i$.

2. Perform this relaxation step until a fixed-point is found:

   a.  Select a spin average $\langle s_i \rangle$ at random from $\langle s \rangle$.

   b.  Compute the mean field $\Phi_i = h_i + 2\sum_{j \neq i} V_{ij} \langle s_j \rangle$.

   c.  Compute the new spin average $\langle s_i \rangle = \{1 + \exp(\Phi_i/T)\}^{-1}$.

3. Decrease $T$ and repeat step 2 until freezing occurs.

**Figure 1.**    The Mean Field Annealing Algorithm

to one while holding the other spin averages constant. By taking the Boltzmann-weighted average of the state values, the spin average is found to be

$$\langle s_i \rangle = \sum_{s_i=0,1} s_i \cdot \frac{\exp(-\Phi_i s_i/T)}{1 + \exp(-\Phi_i/T)} = \frac{1}{1 + \exp(\Phi_i/T)} \ . \tag{3}$$

Equilibrium is established at a given temperature when equations (2) and (3) hold for each spin. The MFA algorithm (Figure 1) begins at a high temperature where this *fixed-point* is easy to determine. The fixed-point is *tracked* as $T$ is lowered by iterating a relaxation step which uses the spin averages to calculate a new mean field that is then used to update the spin averages. As the temperature is lowered, the optimum solution is found as the limit of this sequence of fixed-points.

The relationship of Hopfield neural networks to MFA becomes apparent if the relaxation step in Figure 1 is recast in a *parallel* form in which the entire mean field vector partially moves towards its new state,

$$\Phi_i^{new} = \Phi_i^{old} + \gamma \left( h_i + 2\sum_j V_{ij} \langle s_j \rangle - \Phi_i^{old} \right) \quad \forall i \, ,$$

and then all the spin averages are updated using $\Phi^{new}$. As $\gamma \to 0$, these difference equations become non-linear differential equations,

$$\frac{d\Phi_i}{dt} = h_i + 2\sum_j V_{ij} \langle s_j \rangle - \Phi_i \quad \forall i \, ;$$

which are equivalent to the equations of motion for the Hopfield network (J. J. Hopfield and D. W. Tank (1985)),

$$\frac{du_i}{dt} = \frac{1}{C_i} \left( I_i + \sum_j T_{ij} f(u_j) - \frac{u_i}{\rho_i} \right) \quad \forall i \, ,$$

provided we make $C_i = \rho_i = 1$ and use a sigmoidal transfer function

$$f(u_j) = \frac{1}{1 + \exp(u_j/T)}.$$

Thus, the evolution of a solution in a Hopfield network is a special case of the relaxation toward an equilibrium state effected by the MFA algorithm at a fixed temperature.

## THE GRAPH BIPARTITIONING PROBLEM

Formally, a *graph* consists of a set of $N$ nodes such that nodes $n_i$ and $n_j$ are connected by an edge with weight $V_{ij}$ (which could be zero). The *graph bipartitioning problem* involves equally distributing the graph nodes across two bins, $b_0$ and $b_1$, while minimizing the combined weight of the edges with endpoints in opposite bins. These two sub-objectives tend to frustrate one another in that the first goal is satisfied when the nodes are equally divided between the bins, but the second goal is met (trivially) by assigning all the nodes to a single bin.

## MEAN FIELD FORMULATION

An optimal solution for the bipartitioning problem minimizes the Hamiltonian

$$H = \sum_i \sum_{j \neq i} V_{ij}(1 - s_j)s_i - \tau \sum_i \sum_{j \neq i}(1 - s_j)s_i \quad \text{where } s_i = \begin{cases} 1 & \text{if } n_i \in b_1 \\ 0 & \text{if } n_i \in b_0. \end{cases}$$

In the first term, each edge attracts adjacent nodes into the same bin with a force proportional to its weight. Counterbalancing this attraction is $\tau$, an amorphous repulsive force between all of the nodes which discourages them from clustering together. The average spin of a node $n_i$ can be determined from its mean field:

$$\Phi_i = \sum_{j \neq i}(V_{ij} - \tau) - \sum_{j \neq i} 2(V_{ij} - \tau)\langle s_j \rangle.$$

## EXPERIMENTAL RESULTS

Table 1 compares the performance of the MFA algorithm of Figure 1 with SSA in terms of total optimization and computational effort for 100 trials on each of three example graphs. While the bipartitions found by SSA and MFA are nearly equivalent, MFA required as little as 2% of the number of iterations needed by SSA.

The effect of the decrease in temperature upon the spin averages is depicted in Figure 2. At high temperatures the graph bipartition is maximally disordered, (i.e. $\langle s_i \rangle \approx \frac{1}{2} \ \forall i$), but as the system is cooled past a *critical temperature*, $T_c$, each node

TABLE 1.    Comparison of SSA and MFA on Graph Bipartitioning

|  | $G_1$ | $G_1$ | $G_1$ |
|---|---|---|---|
| Nodes/Edges | 83/115 | 100/200 | 100/400 |
| Solution Value ($H_{MFA}/H_{SSA}$) | 0.762 | 1.078 | 1.030 |
| Relaxation Iterations ($I_{MFA}/I_{SSA}$) | 0.187 | 0.063 | 0.019 |

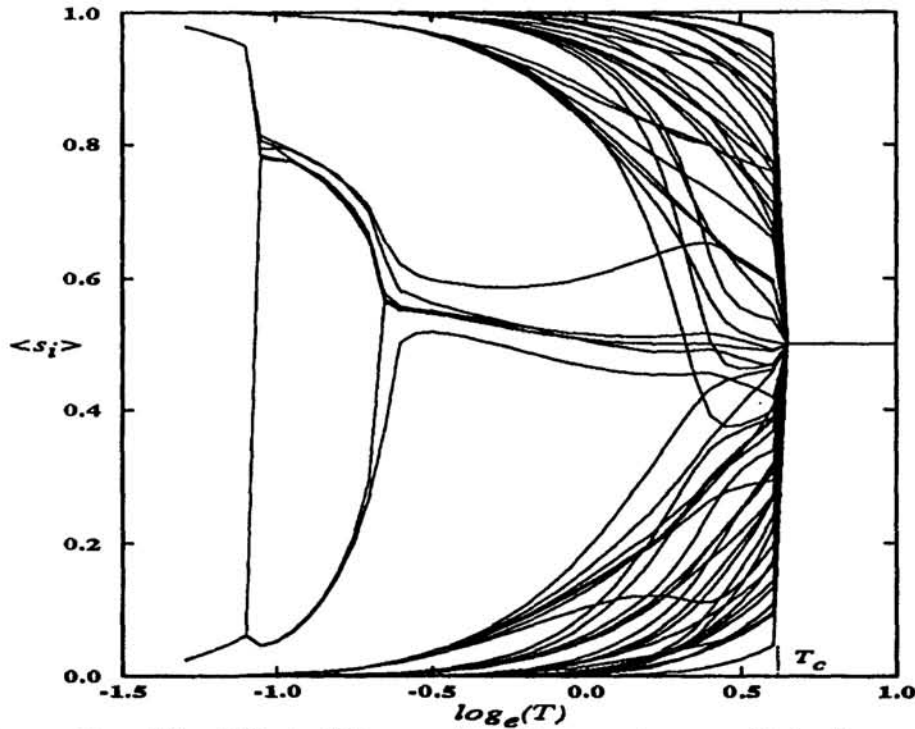

Figure 2.    The Effect of Decreasing Temperature on Spin Averages

begins to move predominantly into one or the other of the two bins (as evidenced by the drift of the spin averages towards 1 or 0). The changes in the spin averages cause $H$ to decrease rapidly in the vicinity of $T_c$.

To analyze the effect of temperature on the spin averages, the behavior of a *cluster* $C$ of spins is idealized with the assumptions:

1.  The repulsive force which balances the bin contents is negligible within $C$ ($r = 0$) compared to the attractive forces arising from the graph edges;

2.  The attractive force exerted by each edge is replaced with an average attractive force $V = \sum_i \sum_j V_{ij}/E$ where $E$ is the number of non-zero weighted edges;

3.  On average, each graph node is adjacent to $\xi = 2E/N$ neighboring nodes;

4.  The movement of the nodes in a cluster can be uniformly described by some deviation, $\sigma$, such that $\langle s_i \rangle = (1 + \sigma)/2$.

Using this model, a cluster moves according to

$$\sigma = \tanh\left(\frac{V\xi}{2T}\sigma\right) . \tag{4}$$

The solution to (4) is a fixed point with $\sigma = 0$ when $T$ is high. This fixed point becomes unstable and the spins diverge from $1/2$ when the temperature is lowered to the point where

$$\left|\frac{d}{d\sigma}\tanh\left(\frac{V\xi}{2T}\sigma\right)\right| > 1 .$$

Solving shows that $T_c = V\xi/2$, which agrees with our experiments and is within $\pm 20\%$ of those observed in (C. Peterson and J. R. Anderson (1987)).

The point at which the nodes *freeze* into their respective bins can be found using (4) and assuming a worst-case situation in which a node is attracted by a single edge (i.e. $\xi = 1$). In this case, the spin deviation will cross an arbitrary threshold, $\sigma_t$ (usually set $\pm 0.9$), when

$$T_f = \frac{V\sigma}{\ln(1 + \sigma_t) - \ln(1 - \sigma_t)} .$$

A cooling schedule is now needed which prescribes how many relaxation iterations, $I_e$, are required at each temperature to reach equilibrium as the system is annealed from $T_c$ to $T_f$. Further analysis of (4) shows that $I_e \propto |T_c/(T_c - T)|$. Thus, more iterations are required to reach equilibrium around $T_c$ than anywhere else, which agrees with observations made during our experiments. The affect of using fewer iterations at various temperatures was empirically studied using the following procedure:

1. Each spin average was initialized to $1/2$ and a small amount of noise was added to break the symmetry of the problem.

2. An initial temperature $T_i$ was imposed, and the mean field equations were iterated $I$ times for each node.

3. After completing the iterations at $T_i$, the temperature was *quenched* to near zero and the mean field equations were again iterated $I$ times to saturate each node at one or zero.

The results of applying this procedure to one of our example graphs with different values of $T_i$ and $I$ are shown in Figure 3. Selecting an initial temperature near $T_c$ and performing sufficient iterations of the mean field equations ($I \geq 40$ in this case) gives final bipartitions that are usually near-optimum, while performing an insufficient number of iterations ($I = 5$ or $I = 20$) leads to poor solutions. However, even a large number of iterations will not compensate if $T_i$ is set so low that the initial convergence causes the graph to abruptly freeze into a local minimum. The highest

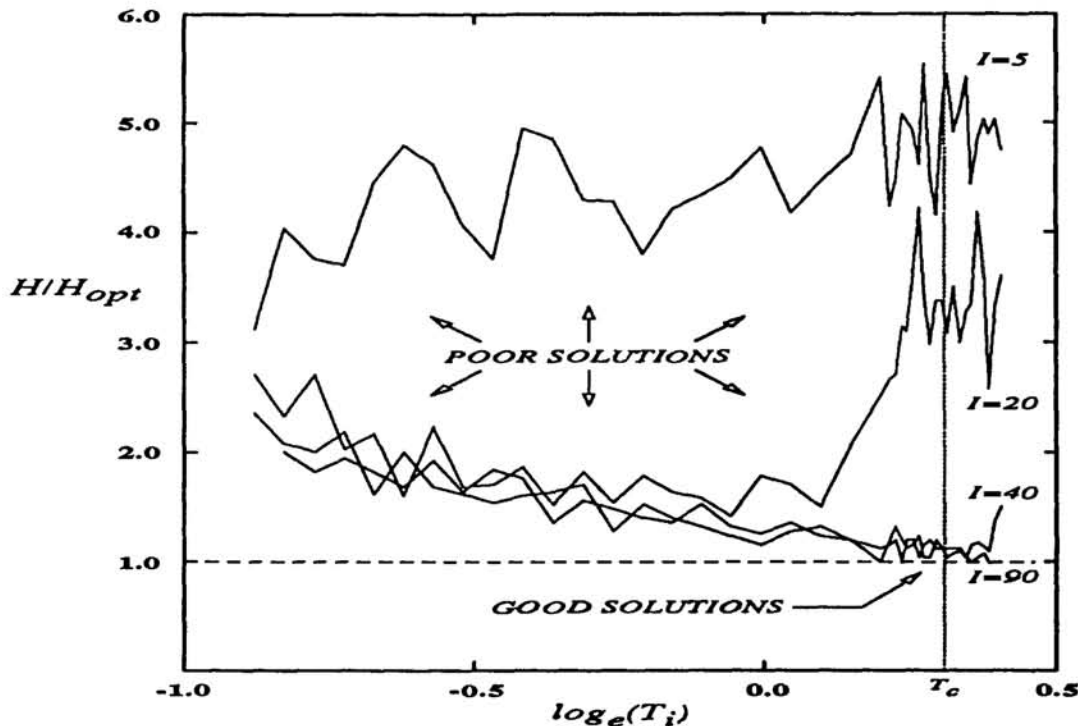

**Figure 3.** The Effect of Initial Temperature and Iterations on the Solution

quality solutions are found when $T_i \approx T_c$ and a sufficient number of relaxations are performed, as shown in the traces for $I = 40$ and $I = 90$. This seems to perform as well as slow cooling and requires much less effort. Obviously, much of the structure of the optimal solution must be present after equilibrating at $T_c$. Due to the equivalence we have shown between Hopfield networks and MFA, this fact may be useful in tuning the gains in Hopfield networks to get better performance.

## CONCLUSIONS

The concept of mean field annealing (MFA) has been introduced and compared to stochastic simulated annealing (SSA) which it closely resembles in both derivation and implementation. In the graph bipartitioning application, we saw the level of optimization achieved by MFA was comparable to that achieved by SSA, but 1-2 orders of magnitude fewer relaxation iterations were required. This speedup is achieved because the average values of the discrete degrees of freedom used by MFA relax to their equilibrium values much faster than the corresponding Markov chain employed in SSA. We have seen similar results when applying MFA to a other problems including $N$-way graph partitioning (D. E. Van den Bout and T. K. Miller III (1988)), restoration of range and luminance images (Griff Bilbro and Wesley Snyder (1988)), and image halftoning (T. K. Miller III and D. E. Van den Bout (1989)). As was shown, the MFA algorithm can be formulated as a parallel iterative procedure, so it should also perform well in parallel processing environments. This has been verified by successfully porting MFA to a ZIP array processor, a 64-node

NCUBE hypercube computer, and a 10-processor Sequent Balance shared-memory multiprocessor with near-linear speedups in each case.

In addition to the speed advantages of MFA, the fact that the system state is represented by continuous variables allows the use of simple analytic techniques to characterize the system dynamics. The dynamics of the MFA algorithm were examined for the problem of graph bipartitioning, revealing the existence of a critical temperature, $T_c$, at which optimization begins to occur. It was also experimentally determined that MFA found better solutions when annealing began near $T_c$ rather than at some lower temperature. Due to the correspondence shown between MFA and Hopfield networks, the critical temperature may be of use in setting the neural gains so that better solutions are found.

## Acknowledgements

This work was partially supported by the North Carolina State University Center for Communications and Signal Processing and Computer Systems Laboratory, and by the Office of Basic Energy Sciences, and the Office of Technology Support Programs, U.S. Department of Energy, under contract No. DE-AC05-84OR21400 with Martin Marietta Energy Systems, Inc.

## References

Griff Bilbro and Wesley Snyder (1988) Image restoration by mean field annealing. In *Advances in Neural Network Information Processing Systems*.

D. E. Van den Bout and T. K. Miller III (1988) Graph partitioning using annealed neural networks. Submitted to *IEEE Trans. on Circuits and Systems*.

J. J. Hopfield and D. W. Tank (1985) Neural computation of decision in optimization problems. *Biological Cybernetics*, **52**, 141–152.

T. K. Miller III and D. E. Van den Bout (1989) Image halftoning by mean field annealing. Submitted to ICNN'89.

S. Kirkpatrick, C. Gelatt, and M. Vecchi (1983) Optimization by simulated annealing. *Science*, **220**(4598), 671–680.

C. Peterson and J. R. Anderson (1987) *Neural Networks and NP-complete Optimization Problems: a Performance Study on the Graph Bisection Problem*. Technical Report MCC-EI-287-87, MCC.

D. J. Thouless, P.W. Anderson, and R.G. Palmer (1977) Solution of 'solvable model of a spin glass'. *Phil. Mag.*, **35**(3), 593–601.